# Image Segmentation with Networks of Variable Scales

Hans P. Graf     Craig R. Nohl     Jan Ben

AT&T Bell Laboratories
Crawfords Corner Road
Holmdel, NJ 07733, USA

## ABSTRACT

We developed a neural net architecture for segmenting complex images, i.e., to localize two-dimensional geometrical shapes in a scene, without prior knowledge of the objects' positions and sizes. A scale variation is built into the network to deal with varying sizes. This algorithm has been applied to video images of railroad cars, to find their identification numbers. Over 95% of the characters were located correctly in a data base of 300 images, despite a large variation in lighting conditions and often a poor quality of the characters. A part of the network is executed on a processor board containing an analog neural net chip (Graf et al. 1991), while the rest is implemented as a software model on a workstation or a digital signal processor.

## 1 INTRODUCTION

Neural nets have been applied successfully to the classification of shapes, such as characters. However, typically, these networks do not tolerate large variations of an object's size. Rather, a normalization of the size has to be done before the network is able to perform a reliable classification. But in many machine vision applications an object's size is not known in advance and may vary over a wide range. If the objects are part of a complex image, finding their positions plus their sizes becomes a very difficult problem.

Traditional techniques to locate objects of variable scale include the generalized Hough transform (Ballard 1981) and constraint search techniques through a feature space (Grimson 1990), possibly with some relaxation mechanisms. These techniques start with a feature representation and then try to sort features into groups that may represent an object. Searches through feature maps tend to be very time consuming, since the number of comparisons that need to be made grows fast, typically exponentially, with the number of features. Therefore, practical techniques must focus on ways to minimize the time required for this search.

Our solution can be viewed as a large network, divided into two parts. The first layer of the network provides a feature representation of the image, while the second layer locates the objects. The key element for this network to be practical, is a neural net chip (Graf et al. 1991) which executes the first layer. The high compute power of this chip makes it possible to extract a large number of features. Hence features specific to the objects to be found can be extracted, reducing drastically the amount of computation required in the second layer.

The output of our network is not necessarily the final solution of a problem. Rather, its intended use is as part of a modular system, combined with other functional elements. Figure 1 shows an example of such a system that was used to read the identification numbers on railroad cars. In this system the network's outputs are the positions and sizes of characters. These are then classified in an additional network (LeCun et al. 1990), specialized for reading characters.

The net described here is not limited to finding characters. It can be combined with other classifiers and is applicable to a wide variety of object recognition tasks. Details of the network, for example the types of features that are extracted, are task specific and have to be optimized for the problem to be solved. But the overall architecture of the network and the data flow remains the same for many problems. Beside the application described here, we used this network for reading the license plates of cars, locating the address blocks on mail pieces, and for page layout analysis of printed documents.

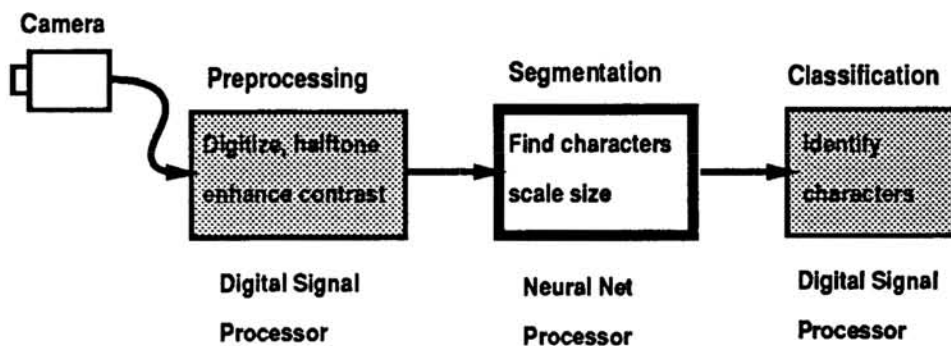

**Figure 1:** Schematic of the recognition system for reading the identification numbers on railroad cars. The network described here performs the part in the middle box, segmenting the image into characters and background.

## 2  THE NETWORK

### 2.1  THE ARCHITECTURE

The network consists of two parts, the input layer extracting features and the second layer, which locates the objects. The second layer is not rigidly coupled through connections to the first one. Before data move from the first layer to the second, the input fields of the neurons in the second layer are scaled to an appropriate size. This size depends on

the data and is dynamically adjusted.

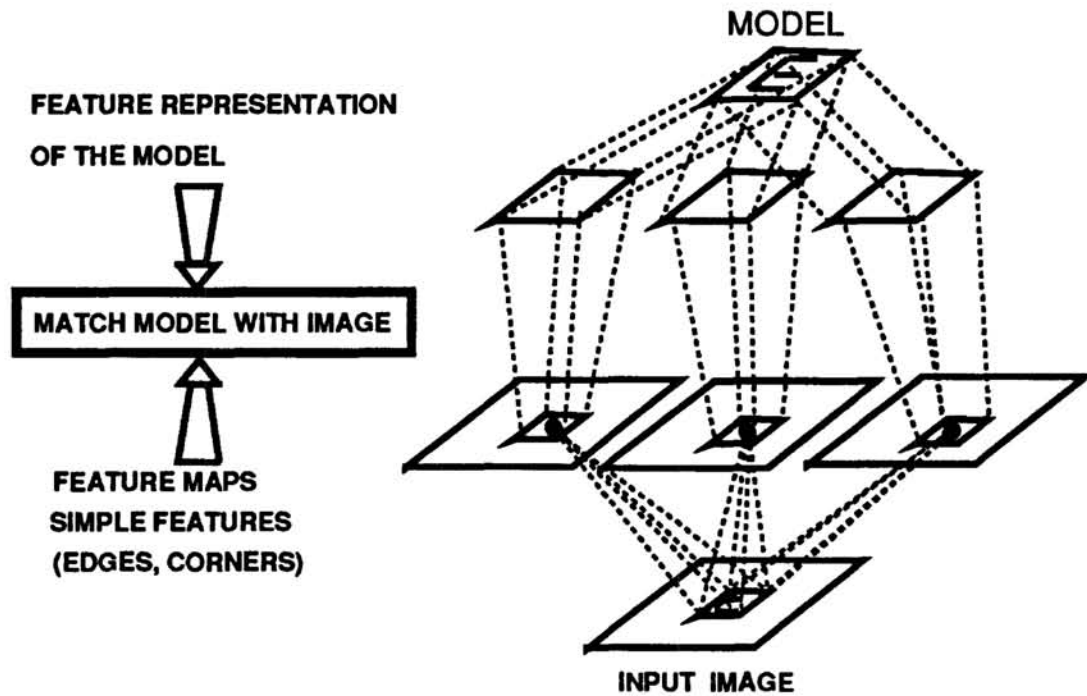

**Figure 2:** Schematic of the network.

Figure 2 shows a schematic of this whole network. The input data pass through the first layer of connections. From the other end of the net the model of the object is entered, and in the middle model and image are matched by scaling the input fields of the neurons in the second layer. In this way a network architecture is obtained that can handle a large variation of sizes. In the present paper we consider only scale variations, but other transformations, such as rotations can be integrated into this architecture as well.

And how can a model representation be scaled to the proper size before one knows an object's size? With a proper feature representation of the image, this can be done in a straight-forward and time-efficient way. Distances between pairs of features are measured and used to scale the input fields. In section 4 it is described in detail how the distances between corners provide a robust estimate of the sizes of characters. There is no need to determine an object's size with absolute certainty here. The goal is to limit the further search to just a few possible sizes, in order to reduce the amount of computation.

The time to evaluate the second layer of the network is reduced further by determining "areas-of-interest" and searching only these. Areas without any features, or without characteristic combinations of features, are excluded from the search. In this way, the neurons of the second layer have to analyze only a small part of the whole image. The key for the size estimates and the "area-of-interest" algorithm to work reliably, is a good feature representation. Thanks to the neural net chip, we can search an image for a large number of geometric features and have great freedom in choosing their shapes.

## 2.2    KERNELS FOR EXTRACTING FEATURES

The features extracted in the first layer have to be detectable regardless of an object's size. Many features, for example corners, are in principle independent of size. In practice however, one uses two-dimensional detectors of a finite extent. These detectors introduce a scale and tend to work best for a certain range of sizes. Hence, it may be necessary to use several detectors of different sizes for one feature. Simple features tend to be less sensitive to scale than complex ones. In the application described below, a variation of a factor of five in the characters' sizes is covered with just a single set of edge and corner detectors. Figure 3 shows a few of the convolution kernels used to extract these features.

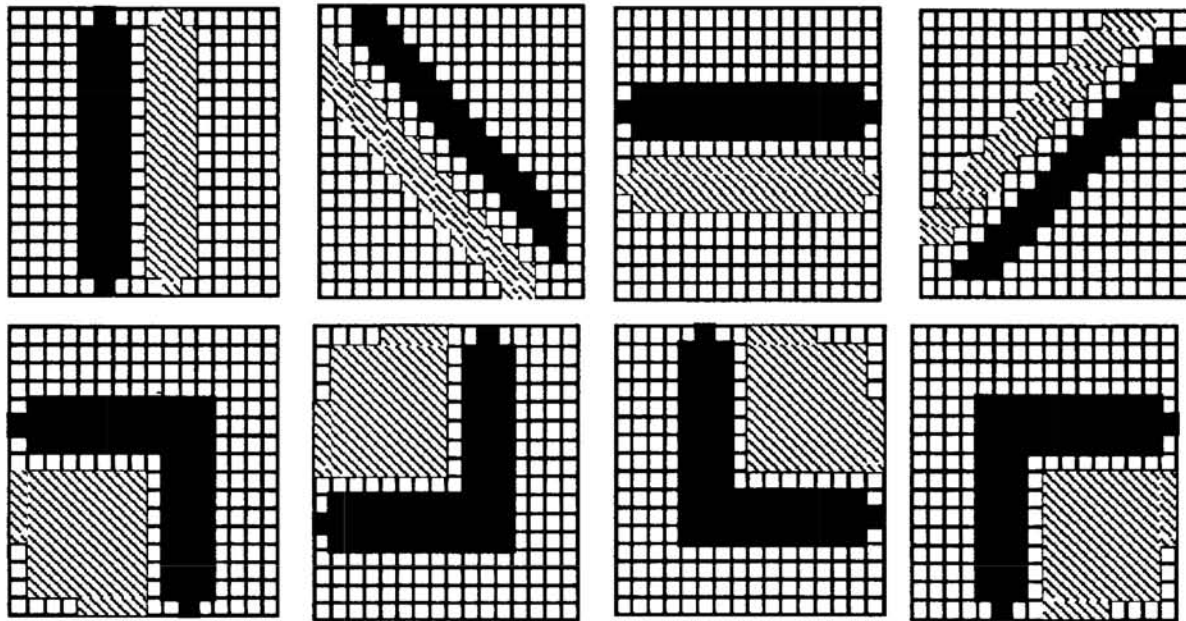

**Figure 3:** Examples of kernels for detecting edges and corners. Each of the kernels is stored as the connection weights of a neuron. These are ternary kernels with a size of 16 x 16 pixels. The values of the pixels are: black = -1, white = 0, hatched = +1. A total of 32 kernels of this size can be scanned simultaneously over an image with the neural net chip.

These kernels are scanned over an image with the neural net chip and wherever an edge or a corner of the proper orientation is located, the neuron tied to this kernel turns on. In this way, the neural net chip scans 32 kernels simultaneously over an image, creating 32 feature maps. The kernels containing the feature detectors have a size of 16 x 16 pixels. With kernels of such a large size, it is possible to create highly selective detectors. More-over, a high noise immunity is obtained.

## 2.3   THE SECOND LAYER

The neurons of the second layer have a rectangular receptive field with 72 inputs, 3 x 3 inputs from eight feature maps. These neurons are trained with feature representations of shapes, normalized in size. The 3 x 3 input field of a neuron does not mean that only an area of 9 pixels in a feature map is used as input. Before a neuron is scanned over a part of a feature map, its input field is scaled to the size indicated by the size estimator. Therefore, each input corresponds to a rectangular area in a feature map. For finding objects in an image, the input fields, scaled to the proper size, are then scanned over the areas marked by the "area-of-interest" algorithm. If an output of a neuron is high, the area is marked as position of an object and is passed along to the classifier.

The second layer of the network does require only relatively few computations, typically a few hundred evaluations of neurons with 72 inputs. Therefore, this can be handled easily by a workstation or a digital signal processor. The same is true for the area-of-interest algorithm. The computationally expensive part is the feature extraction. On an image with 512 x 512 pixels this requires over 2 billion connection updates. In fact, on a workstation this takes typically about half an hour. Therefore, here a special purpose chip is crucial to provide a speed-up to make this approach useful for practical applications.

## 3   THE HARDWARE

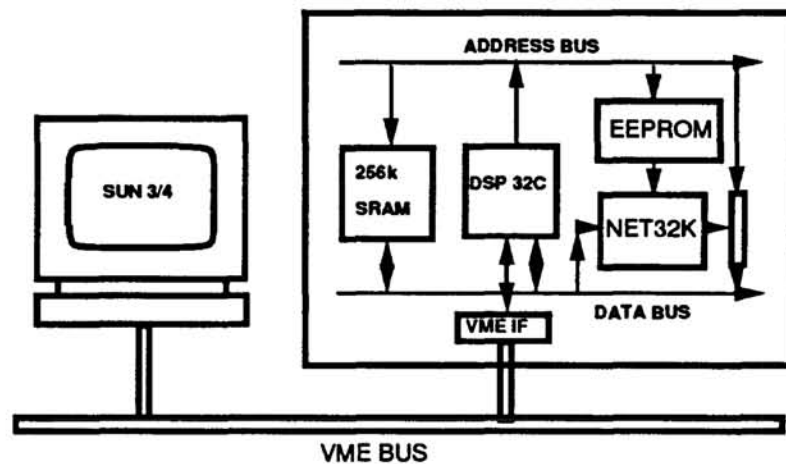

**Figure 4:** Schematic of the neural net board.

A schematic of the neural net board used for these applications is shown in Figure 4. The board contains an analog neural net chip, combined with a digital signal processor (DSP) and 256k of fast static memory. On the board, the DSP controls the data flow and the operation of the neural net chip. This board is connected over a VME bus to the host workstation. Signals, such as images, are sent from the host to the neural net board, where a local program operates on the data. The results are then sent back to the host for further processing and display. The time it takes to process an image of 512 x 512 pixels is one second, where the transfer of the data from the workstation to the board and back requires two thirds of this time.

The chip does a part of the computation in analog form. But analog signals are used only inside the chip, while all the input and the output data are digital. This chip works only with a low dynamic range of the signals. Therefore, the input signals are typically binarized before they are transferred to the chip. In the case of gray level images, the pictures are halftoned first and then the features are extracted form the binarized images. This is possible, since the large kernel sizes suppress the noise introduced by the halftoning process.

# 4  APPLICATION

This network was integrated into a system to read the identification numbers on railroad cars. Identifying a rail car by its number has to be done before a train enters the switching yard, where the cars are directed to different tracks. Today this is handled by human operators reading the numbers from video screens. The present investigation is a study to determine whether this process can be automated.

The pictures represent very difficult segmentation tasks, since the size of the characters varies by more than a factor of five and they are often of poor quality with parts rusted away or covered by dirt. Moreover, the positions of the characters can be almost anywhere in the picture, and they may be arranged in various ways, in single or in multiple lines. Also, they are written in many different fonts, and the contrast between characters and background varies substantially from one car to the next. Despite these difficulties, we were able to locate the characters correctly in over 95% of the cases, on a database of 300 video images of railroad cars.

As mentioned in section 2, in the first layer feature maps are created from which areas of interest are determined. Since the characters are arranged in horizontal lines, the first step is to determine where lines of characters might be present in the image. For that purpose the feature maps are projected onto a vertical line. Rows of characters produce strong responses of the corner detectors and are therefore detected as maxima in the projected densities. The orientation of a corner indicates whether it resulted from the lower end of a character or from the upper end. The simultaneous presence of maxima in the densities of lower and upper ends is therefore a strong indication for the presence of a row of characters. In this way, bands within the image are identified that may contain characters. A band not only indicates the presence of a row of characters, but also provides a good guess of their heights.

This simple heuristic proved to be very effective for the rail car images. It was made more robust by taking into account also the outputs of the vertical edge detectors. Characters produce strong responses of vertical edge detectors, while detractors, such as dirt create fewer and weaker responses.

At this stage we do not attempted to identify a character. All we need is a yes/no answer whether a part of the image should be analyzed by the classifier or not. The whole alphabet is grouped into five classes, and only one neuron to recognize any member within a class is created. A high output of one of these neurons therefore means that any character of its class may be present. Figures 5 and 6 show two examples produced by the segmentation network. The time required for the whole segmentation is less than three seconds, of which one second is spent for the feature extraction and the rest for the "focus-of-

attention" algorithm and the second layer of the network.

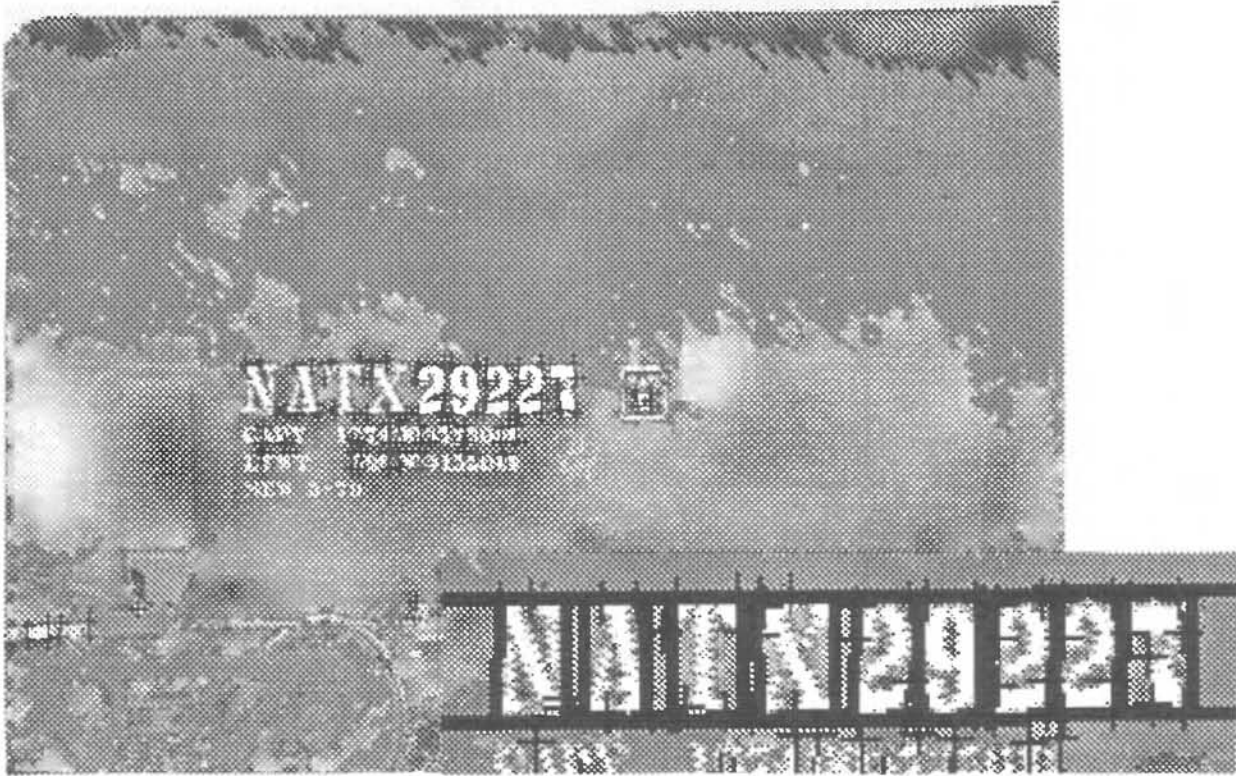

**Figure 5:** Image of a tank car. The crosses mark where corner detectors gave a strong response. The inset shows an enlarged section around the identification number. The result of the segmentation network is indicated by the black lines.

## 5  CONCLUSION

The algorithm described combines neural net techniques with heuristics to obtain a practical solution for segmenting complex images reliably and fast. Clearly, a "conventional" neural net with a fixed architecture lacks the flexibility to handle the scale variations required in many machine vision applications. To extend the use of neural nets, transformations have to be built into the architecture.

We demonstrated the network's use for locating characters, but the same strategy works for a wide variety of other objects. Some details need to be adjusted to the objects to be found. In particular, the features extracted by the first layer are task specific. Their choice is critical, as they determine to a large extent the computational requirements for finding the objects in the second layer.

The use of a neural net chip is crucial to make this approach feasible, since it provides the computational power needed for the feature extraction. The extraction of geometrical features for pattern recognition applications has been studied extensively. However, its use is not wide spread, since it is computationally very demanding. The neural net chip opens the possibility for extracting large numbers of features in a short time. The large size of the convolution kernels, 16 x 16 pixels, provides a great flexibility in choosing the feature detectors' shapes. Their large size is also the main reason for a good noise

suppression and a high robustness of the described network.

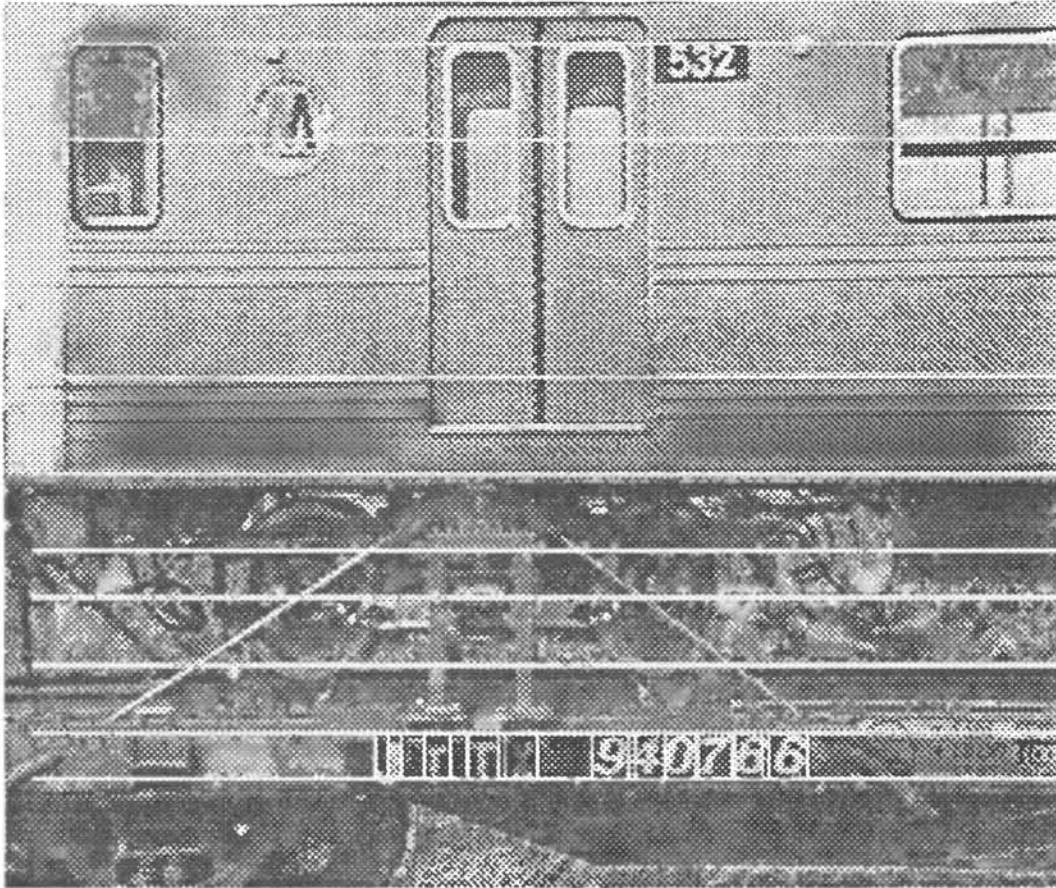

**Figure 6:** The result of the network on an image of high complexity. The white horizontal lines indicate the result of the "area-of-interest" algorithm. The final result is shown by the vertical white lines.

## References

H.P. Graf, R. Janow, C.R. Nohl, and J. Ben, (1991), "A Neural-Net Board System for Machine Vision Applications", *Proc. Int. Joint Conf. Neural Networks*, Vol. 1, pp. 481 - 486.

D.H. Ballard, (1981), "Generalizing the Hough transform to detect arbitrary shapes", *Pattern Recognition*, Vol. 13, p. 111.

W.H. Grimson, (1990), "The Combinatorics of Object Recognition in Cluttered Environments Using Constraint Search", *Artificial Intelligence*, Vol. 44, p. 121.

Y. LeCun, B. Boser, J.S. Denker, D. Henderson, R.E. Howard, W. Hubbard, and L.D. Jackel, (1990), "Handwritten Digit Recognition with a Back-Propagation Network", in: Neural Information Processing Systems, Vol. 2, D. Touretzky (ed.), Morgan Kaufman, pp. 396 - 404.
